# Kernel Design Using Boosting

**Koby Crammer   Joseph Keshet   Yoram Singer**
School of Computer Science & Engineering
The Hebrew University, Jerusalem 91904, Israel
{kobics,jkeshet,singer}@cs.huji.ac.il

## Abstract

The focus of the paper is the problem of learning kernel operators from empirical data. We cast the kernel design problem as the construction of an accurate kernel from simple (and less accurate) base kernels. We use the boosting paradigm to perform the kernel construction process. To do so, we modify the booster so as to accommodate kernel operators. We also devise an efficient weak-learner for simple kernels that is based on generalized eigen vector decomposition. We demonstrate the effectiveness of our approach on synthetic data and on the USPS dataset. On the USPS dataset, the performance of the Perceptron algorithm with learned kernels is systematically better than a fixed RBF kernel.

## 1   Introduction and problem Setting

The last decade brought voluminous amount of work on the design, analysis and experimentation of kernel machines. Algorithm based on kernels can be used for various machine learning tasks such as classification, regression, ranking, and principle component analysis. The most prominent learning algorithm that employs kernels is the Support Vector Machines (SVM) [1, 2] designed for classification and regression. A key component in a kernel machine is a *kernel operator* which computes for any pair of instances their inner-product in some abstract vector space. Intuitively and informally, a kernel operator is a means for measuring similarity between instances. Almost all of the work that employed kernel operators concentrated on various machine learning problems that involved a *predefined* kernel. A typical approach when using kernels is to choose a kernel before learning starts. Examples to popular predefined kernels are the Radial Basis Functions and the polynomial kernels (see for instance [1]). Despite the simplicity required in modifying a learning algorithm to a "kernelized" version, the success of such algorithms is not well understood yet. More recently, special efforts have been devoted to crafting kernels for specific tasks such as text categorization [3] and protein classification problems [4].

Our work attempts to give a computational alternative to predefined kernels by learning kernel operators from data. We start with a few definitions. Let $\mathcal{X}$ be an instance space. A kernel is an inner-product operator $K : \mathcal{X} \times \mathcal{X} \rightarrow \mathbb{R}$. An explicit way to describe $K$ is via a mapping $\phi : \mathcal{X} \rightarrow \mathcal{H}$ from $\mathcal{X}$ to an inner-products space $\mathcal{H}$ such that $K(x, x') = \phi(x) \cdot \phi(x')$. Given a kernel operator and a finite set of instances $S = \{x_i, y_i\}_{i=1}^m$, the kernel matrix (a.k.a the Gram matrix) is the matrix of all possible inner-products of pairs from $S$, $K_{i,j} = K(x_i, x_j)$. We therefore refer to the general form of $K$ as the kernel *operator* and to the application of the kernel operator to a set of pairs of instances as the kernel *matrix*.

The specific setting of kernel design we consider assumes that we have access to a *base kernel learner* and we are given a *target* kernel $K^\star$ manifested as a kernel matrix on a set of examples. Upon calling the base kernel learner it returns a kernel operator denote $K_j$. The goal thereafter is to find a weighted combination of kernels $\hat{K}(x, x') = \sum_j \alpha_j K_j(x, x')$ that is similar, in a sense that will be defined shortly, to the target kernel, $\hat{K} \sim K^\star$. Cristianini et al. [5] in their pioneering work on kernel target alignment employed as the notion of similarity the inner-product between the kernel matrices $< K, K' >_F = \sum_{i,j=1}^m K(x_i, x_j)K'(x_i, x_j)$. Given this definition, they defined the kernel-similarity, or alignment, to be the above inner-product normalized by the norm of each kernel, $\hat{A}(S, \hat{K}, K^\star) = \left( < \hat{K}, K^\star >_F \right) / \sqrt{< \hat{K}, \hat{K} >_F < K^\star, K^\star >_F}$, where $S$ is, as above, a finite sample of $m$ instances. Put another way, the kernel alignment Cristianini et al. employed is the cosine of the angle between the kernel matrices where each matrix is "flattened" into a vector of dimension $m^2$. Therefore, this definition implies that the alignment is bounded above by 1 and can attain this value iff the two kernel matrices are identical. Given a (column) vector of $m$ labels $y$ where $y_i \in \{-1, +1\}$ is the label of the instance $x_i$, Cristianini et al. used the outer-product of $y$ as the the target kernel, $K^\star = yy^T$. Therefore, an optimal alignment is achieved if $\hat{K}(x_i, x_j) = y_i y_j$. Clearly, if such a kernel is used for classifying instances from $\mathcal{X}$, then the kernel itself suffices to construct an excellent classifier $f : \mathcal{X} \to \{-1, +1\}$ by setting, $f(x) = \text{sign}(y_i K(x_i, x))$ where $(x_i, y_i)$ is any instance-label pair. Cristianini et al. then devised a procedure that works with both labelled and unlabelled examples to find a *Gram matrix* which attains a good alignment with $K^\star$ on the labelled part of the matrix. While this approach can clearly construct powerful kernels, a few problems arise from the notion of kernel alignment they employed. For instance, a kernel operator such that the $\text{sign}(K(x_i, x_j))$ is equal to $y_i y_j$ but its magnitude, $|K(x_i, x_j)|$, is not necessarily 1, might achieve a poor alignment score while it can constitute a classifier whose empirical loss is zero. Furthermore, the task of finding a good kernel when it is not always possible to find a kernel whose sign on each pair of instances is equal to the products of the labels (termed the soft-margin case in [5, 6]) becomes rather tricky. We thus propose a different approach which attempts to overcome some of the difficulties above.

Like Cristianini et al. we assume that we are given a set of labelled instances $S = \{(x_i, y_i) \mid x_i \in \mathcal{X}, \ y_i \in \{-1, +1\}, \ i = 1, \ldots, m \}$. We are also given a set of unlabelled examples $\tilde{S} = \{\tilde{x}_i\}_{i=1}^{\tilde{m}}$. If such a set is not provided we can simply use the labelled instances (without the labels themselves) as the set $\tilde{S}$. The set $\tilde{S}$ is used for constructing the primitive kernels that are combined to constitute the learned kernel $\hat{K}$. The labelled set is used to form the target kernel matrix and its instances are used for evaluating the learned kernel $\hat{K}$. This approach, known as transductive learning, was suggested in [5, 6] for kernel alignment tasks when the distribution of the instances in the test data is different from that of the training data. This setting becomes in particular handy in datasets where the test data was collected in a different scheme than the training data. We next discuss the notion of kernel goodness employed in this paper. This notion builds on the objective function that several variants of boosting algorithms maintain [7, 8]. We therefore first discuss in brief the form of boosting algorithms for kernels.

## 2  Using Boosting to Combine Kernels

Numerous interpretations of AdaBoost and its variants cast the boosting process as a procedure that attempts to minimize, or make small, a continuous bound on the classification error (see for instance [9, 7] and the references therein). A recent work by Collins et al. [8] unifies the boosting process for two popular loss functions, the exponential-loss (denoted henceforth as ExpLoss) and logarithmic-loss (denoted as LogLoss) that bound the empir-

**Input:** Labelled and unlabelled sets of examples: $S = \{(x_i, y_i)\}_{i=1}^m$ ; $\tilde{S} = \{\tilde{x}_i\}_{i=1}^{\tilde{m}}$
**Initialize:** $K \leftarrow \mathbf{0}$ (all zeros matrix)
**For** $t = 1, 2, \ldots, T$:

- Calculate distribution over pairs $1 \leq i, j \leq m$:

$$D_t(i,j) = \begin{cases} \exp(-y_i y_j K(x_i, x_j)) & \text{ExpLoss} \\ 1/(1 + \exp(-y_i y_j K(x_i, x_j))) & \text{LogLoss} \end{cases}$$

- Call `base-kernel-learner` with $(D_t, S, \tilde{S})$ and receive $K_t$
- Calculate:

$S_t^+ = \{(i,j) \mid y_i y_j K_t(x_i, x_j) > 0\}$ ; $S_t^- = \{(i,j) \mid y_i y_j K_t(x_i, x_j) < 0\}$
$W_t^+ = \sum_{(i,j) \in S_t^+} D_t(i,j) |K_t(x_i, x_j)|$ ; $W_t^- = \sum_{(i,j) \in S_t^-} D_t(i,j) |K_t(x_i, x_j)|$

- Set: $\alpha_t = \frac{1}{2} \ln \left( \frac{W_t^+}{W_t^-} \right)$ ; $K \leftarrow K + \alpha_t K_t$.

**Return:** kernel operator $K : \mathcal{X} \times \mathcal{X} \to \mathbb{R}$

Figure 1: The skeleton of the boosting algorithm for kernels.

ical classification error. Given the prediction of a classifier $f$ on an instance $x$ and a label $y \in \{-1, +1\}$ the ExpLoss and the LogLoss are defined as,

$$\text{ExpLoss}(f(x), y) = \exp(-yf(x))$$
$$\text{LogLoss}(f(x), y) = \log(1 + \exp(-yf(x))) \ .$$

Collins et al. described a single algorithm for the two losses above that can be used within the boosting framework to construct a strong-hypothesis which is a classifier $f(x)$. This classifier is a weighted combination of (possibly very simple) base classifiers. (In the boosting framework, the base classifiers are referred to as weak-hypotheses.) The strong-hypothesis is of the form $f(x) = \sum_{t=1}^T \alpha_t h_t(x)$. Collins et al. discussed a few ways to select the weak-hypotheses $h_t$ and to find a good of weights $\alpha_t$. Our starting point in this paper is the first *sequential* algorithm from [8] that enables the construction or creation of weak-hypotheses on-the-fly. We would like to note however that it is possible to use other variants of boosting to design kernels.

In order to use boosting to design kernels we extend the algorithm to operate over pairs of instances. Building on the notion of alignment from [5, 6], we say that the inner-product of $x_1$ and $x_2$ is aligned with the labels $y_1$ and $y_2$ if $\text{sign}(K(x_1, x_2)) = y_1 y_2$. Furthermore, we would like to make the magnitude of $K(x, x')$ to be as large as possible. We therefore use one of the following two alignment losses for a pair of examples $(x_1, y_1)$ and $(x_2, y_2)$,

$$\text{ExpLoss}(K(x_1, x_2), y_1 y_2) = \exp(-y_1 y_2 K(x_1, x_2))$$
$$\text{LogLoss}(K(x_1, x_2), y_1 y_2) = \log(1 + \exp(-y_1 y_2 K(x_1, x_2))) \ .$$

Put another way, we view a *pair* of instances as a single example and cast the pairs of instances that attain the same label as positively labelled examples while pairs of opposite labels are cast as negatively labelled examples. Clearly, this approach can be applied to both losses. In the boosting process we therefore maintain a distribution over pairs of instances. The weight of each pair reflects how difficult it is to predict whether the labels of the two instances are the same or different. The core boosting algorithm follows similar lines to boosting algorithms for classification algorithm. The pseudo code of the booster is given in Fig. 1. The pseudo-code is an adaptation the to problem of kernel design of the sequential-update algorithm from [8]. As with other boosting algorithm, the base-learner, which in our case is charge of returning a good kernel with respect to the current distribution, is left unspecified. We therefore turn our attention to the algorithmic implementation of the base-learning algorithm for kernels.

# 3  Learning Base Kernels

The base kernel learner is provided with a training set $S$ and a distribution $D_t$ over a pairs of instances from the training set. It is also provided with a set of unlabelled examples $\tilde{S}$. Without any knowledge of the topology of the space of instances a learning algorithm is likely to fail. Therefore, we assume the existence of an *initial* inner-product over the input space. We assume for now that this initial inner-product is the standard scalar products over vectors in $\mathbb{R}^n$. We later discuss a way to relax the assumption on the form of the inner-product. Equipped with an inner-product, we define the family of base kernels to be the possible *outer-products* $K_w = ww^T$ between a vector $w \in \mathbb{R}^n$ and itself.

Using this definition we get,

$K_w(x_i, x_j) = (x_i \cdot w)(x_j \cdot w)$.

Therefore, the similarity between two instances $x_i$ and $x_j$ is high iff both $x_i$ and $x_j$ are similar (w.r.t the standard inner-product) to a third vector $w$. Analogously, if both $x_i$ and $x_j$ seem to be dissimilar to the vector $w$ then they are similar to each other. Despite the restrictive form of the inner-products, this family is still too rich for our setting and we further impose two restrictions on the inner products. First, we assume

**Input:** A distribution $D_t$. Labelled and unlabelled sets: $S = \{(x_i, y_i)\}_{i=1}^m$ ; $\tilde{S} = \{\tilde{x}_i\}_{i=1}^{\tilde{m}}$ .
**Compute** :

- Calculate:
  $A \in \mathbb{R}^{m \times \tilde{m}}$ , $A_{i,r} = x_i \cdot \tilde{x}_r$
  $B \in \mathbb{R}^{m \times m}$ , $B_{i,j} = D_t(i,j) y_i y_j$
  $K \in \mathbb{R}^{\tilde{m} \times \tilde{m}}$ , $K_{r,s} = \tilde{x}_r \cdot \tilde{x}_s$
- Find the generalized eigenvector $v \in \mathbb{R}^m$ for the problem $A^T B A v = \lambda K v$ which attains the largest eigenvalue $\lambda$
- Set: $w = (\sum_r v_r \tilde{x}_r)/\|\sum_r v_r \tilde{x}_r\|$.

**Return:** Kernel operator $K_w = ww^t$.

Figure 2: The base kernel learning algorithm.

that $w$ is restricted to a linear combination of vectors from $\tilde{S}$. Second, since scaling of the base kernels is performed by the boosted, we constrain the norm of $w$ to be 1. The resulting class of kernels is therefore, $\mathcal{C} = \{K_w = ww^T \mid w = \sum_{r=1}^{\tilde{m}} \beta_r \tilde{x}_r, \|w\| = 1\}$ . In the boosting process we need to choose a specific base-kernel $K_w$ from $\mathcal{C}$. We therefore need to devise a notion of how good a candidate for base kernel is given a labelled set $S$ and a distribution function $D_t$. In this work we use the simplest version suggested by Collins et al. This version can been viewed as a linear approximation on the loss function. We define the score of a kernel $K_w$ w.r.t to the current distribution $D_t$ to be,

$$\text{Score}(K_w) = \sum_{i,j} D_t(i,j) y_i y_j K_w(x_i, x_j) . \tag{1}$$

The higher the value of the score is, the better $K_w$ fits the training data. Note that if $D_t(i,j) = 1/m^2$ (as is $D_0$) then $\text{Score}(K_w)$ is proportional to the alignment since $\|w\| = 1$. Under mild assumptions the score can also provide a lower bound of the loss function. To see that let $c$ be the derivative of the loss function at margin zero, $c = |\text{Loss}'(0)|$. If all the training examples $x_i \in S$ lies in a ball of radius $\sqrt{c}$, we get that $\text{Loss}(K_w(x_i, x_j), y_i y_j) \geq 1 - cK_w(x_i, x_j) y_i y_j \geq 0$, and therefore,

$$\sum_{i,j} D_t(i,j) \text{Loss}(K_w(x_i, x_j), y_i y_j) \geq 1 - c \sum_{i,j} D_t(i,j) K_w(x_i, x_j) y_i y_j .$$

Using the explicit form of $K_w$ in the Score function (Eq. (1)) we get, $\text{Score}(K_w) = \sum_{i,j} D(i,j) y_i y_j (w \cdot x_i)(w \cdot x_j)$ . Further developing the above equation using the constraint that $w = \sum_{r=1}^{\tilde{m}} \beta_r \tilde{x}_r$ we get,

$$\text{Score}(K_w) = \sum_{r,s} \beta_s \beta_r \sum_{i,j} D(i,j) y_i y_j (x_i \cdot \tilde{x}_r)(x_j \cdot \tilde{x}_s) .$$

To compute efficiently the base kernel score *without* an explicit enumeration we exploit the fact that if the initial distribution $D_0$ is symmetric ($D_0(i,j) = D_0(j,i)$) then all the distributions generated along the run of the boosting process, $D_t$, are also symmetric. We now define a matrix $A \in \mathbb{R}^{m \times \tilde{m}}$ where $A_{i,r} = x_i \cdot \tilde{x}_r$ and a symmetric matrix $B \in \mathbb{R}^{m \times m}$ with $B_{i,j} = D_t(i,j)y_i y_j$. Simple algebraic manipulations yield that the score function can be written as the following quadratic form, $\text{Score}(\beta) = \beta^T (A^T B A)\beta$ , where $\beta$ is $\tilde{m}$ dimensional column vector. Note that since $B$ is symmetric so is $A^T B A$. Finding a good base kernel is equivalent to finding a vector $\beta$ which *maximizes* this quadratic form under the norm equality constraint $\|w\|^2 = \|\sum_{r=1}^{\tilde{m}} \beta_r \tilde{x}_r\|^2 = \beta^T K \beta = 1$ where $K_{r,s} = \tilde{x}_r \cdot \tilde{x}_s$ . Finding the maximum of $\text{Score}(\beta)$ subject to the norm constraint is a well known maximization problem known as the generalized eigen vector problem (cf. [10]). Applying simple algebraic manipulations it is easy to show that the matrix $A^T B A$ is positive semi-definite. Assuming that the matrix $K$ is invertible, the the vector $\beta$ which maximizes the quadratic form is proportional the eigenvector of $K^{-1}A^T B A$ which is associated with the generalized largest eigenvalue. Denoting this vector by $v$ we get that $w \propto \sum_{r=1}^{\tilde{m}} v_r \tilde{x}_r$. Adding the norm constraint we get that $w = (\sum_{r=1}^{\tilde{m}} v_r \tilde{x}_r)/\|\sum_{r=1}^{\tilde{m}} v_r \tilde{x}_r\|$. The skeleton of the algorithm for finding a base kernels is given in Fig. 3. To conclude the description of the kernel learning algorithm we describe how to the extend the algorithm to be employed with general kernel functions.

**Kernelizing the Kernel:** As described above, we assumed that the standard scalar-product constitutes the template for the class of base-kernels $\mathcal{C}$. However, since the procedure for choosing a base kernel depends on $S$ and $\tilde{S}$ only through the inner-products matrix $A$, we can replace the scalar-product itself with a general kernel operator $\kappa : \mathcal{X} \times \mathcal{X} \to \mathbb{R}$, where $\kappa(x_i, x_j) = \phi(x_i) \cdot \phi(x_j)$. Using a general kernel function $\kappa$ we can not compute however the vector $w$ explicitly. We therefore need to show that the norm of $w$, and evaluation $K_w$ on any two examples can still be performed efficiently.

First note that given the vector $v$ we can compute the norm of $w$ as follows,

$$\|w\|^2 = \left(\sum_r v_r \tilde{x}_r\right)^T \left(\sum_s v_s \tilde{x}_r\right) = \sum_{r,s} v_r v_s \kappa(\tilde{x}_r, \tilde{x}_s) \ .$$

Next, given two vectors $x_i$ and $x_j$ the value of their inner-product is,

$$K_w(x_i, x_j) = \sum_{r,s} v_r v_s \kappa(x_i, \tilde{x}_r)\kappa(x_j, \tilde{x}_s) \ .$$

Therefore, although we cannot compute the vector $w$ explicitly we can still compute its norm and evaluate any of the kernels from the class $\mathcal{C}$.

## 4 Experiments

**Synthetic data:** We generated binary-labelled data using as input space the vectors in $\mathbb{R}^{100}$. The labels, in $\{-1, +1\}$, were picked uniformly at random. Let $y$ designate the label of a particular example. Then, the first two components of each instance were drawn from a two-dimensional normal distribution, $\mathcal{N}(\mu, \Delta \sum \Delta^{-1})$ with the following parameters,

$$\mu = y \begin{pmatrix} 0.03 \\ 0.03 \end{pmatrix} \qquad \Delta = \frac{1}{\sqrt{2}} \begin{pmatrix} 1 & -1 \\ 1 & 1 \end{pmatrix} \qquad \sum = \begin{pmatrix} 0.1 & 0 \\ 0 & 0.01 \end{pmatrix} .$$

That is, the label of each examples determined the mean of the distribution from which the first two components were generated. The rest of the components in the vector (98

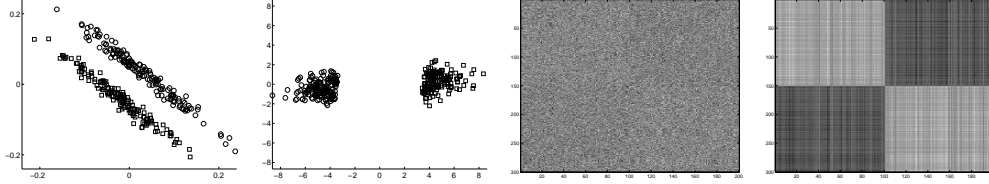

Figure 3: Results on a toy data set prior to learning a kernel (first and third from left) and after learning (second and fourth). For each of the two settings we show the first two components of the training data (left) and the matrix of inner products between the train and the test data (right).

altogether) were generated independently using the normal distribution with a zero mean and a standard deviation of $0.05$. We generated 100 training and test sets of size 300 and 200 respectively. We used the standard dot-product as the initial kernel operator.

On each experiment we first learned a linear classier that separates the classes using the Perceptron [11] algorithm. We ran the algorithm for 10 epochs on the training set. After each epoch we evaluated the performance of the current classifier on the test set. We then used the boosting algorithm for kernels with the LogLoss for 30 rounds to build a kernel for each random training set. After learning the kernel we re-trained a classifier with the Perceptron algorithm and recorded the results. A summary of the online performance is given in Fig. 4. The plot on the left-hand-side of the figure shows the instantaneous error (achieved during the run of the algorithm). Clearly, the Perceptron algorithm with the learned kernel converges much faster than the original kernel. The middle plot shows the test error after each epoch. The plot on the right shows the test error on a noisy test set in which we added a Gaussian noise of zero mean and a standard deviation of $0.03$ to the first two features. In all plots, each bar indicates a $95\%$ confidence level. It is clear from the figure that the original kernel is much slower to converge than the learned kernel. Furthermore, though the kernel learning algorithm was not expoed to the test set noise, the learned kernel reflects better the structure of the feature space which makes the learned kernel more robust to noise.

Fig. 3 further illustrates the benefits of using a boutique kernel. The first and third plots from the left correspond to results obtained using the original kernel and the second and fourth plots show results using the learned kernel. The left plots show the empirical distribution of the two informative components on the test data. For the learned kernel we took each input vector and projected it onto the two eigenvectors of the learned kernel operator matrix that correspond to the two largest eigenvalues. Note that the distribution after the projection is bimodal and well separated along the first eigen direction ($x$-axis) and shows rather little deviation along the second eigen direction ($y$-axis). This indicates that the kernel learning algorithm indeed found the most informative projection for separating the labelled data with large margin. It is worth noting that, in this particular setting, any algorithm which chooses a single feature at a time is prone to failure since both the first and second features are mandatory for correctly classifying the data.

The two plots on the right hand side of Fig. 3 use a gray level color-map to designate the value of the inner-product between each pairs instances, one from training set ($y$-axis) and the other from the test set. The examples were ordered such that the first group consists of the positively labelled instances while the second group consists of the negatively labelled instances. Since most of the features are non-relevant the original inner-products are noisy and do not exhibit any structure. In contrast, the inner-products using the learned kernel yields in a $2 \times 2$ block matrix indicating that the inner-products between instances sharing the same label obtain large positive values. Similarly, for instances of opposite

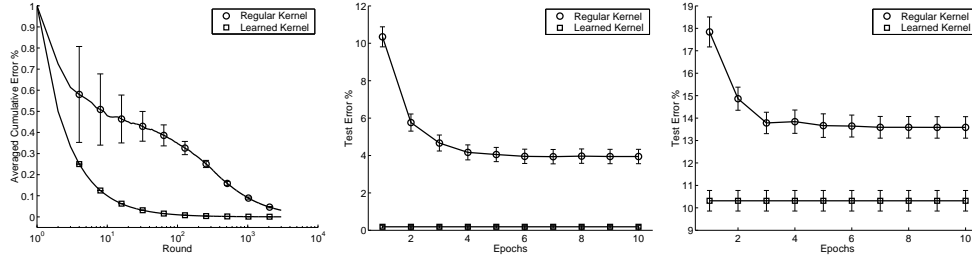

Figure 4: The online training error (left), test error (middle) on clean synthetic data using a standard kernel and a learned kernel. Right: the online test error for the two kernels on a noisy test set.

labels the inner products are large and negative. The form of the inner-products matrix of the learned kernel indicates that the learning problem itself becomes much easier. Indeed, the Perceptron algorithm with the standard kernel required around $94$ training examples on the average before converging to a hyperplane which perfectly separates the training data while using the Perceptron algorithm with learned kernel required a *single* example to reach a perfect separation on all $100$ random training sets.

**USPS dataset:** The USPS (US Postal Service) dataset is known as a challenging classification problem in which the training set and the test set were collected in a different manner. The USPS contains $7,291$ training examples and $2,007$ test examples. Each example is represented as a $16 \times 16$ matrix where each entry in the matrix is a pixel that can take values in $\{0, \ldots, 255\}$. Each example is associated with a label in $\{0, \ldots, 9\}$ which is the digit content of the image. Since the kernel learning algorithm is designed for binary problems, we broke the 10-class problem into $45$ binary problems by comparing all pairs of classes. The interesting question of how to learn kernels for multiclass problems is beyond the scopre of this short paper. We thus constraint on the binary error results for the $45$ binary problem described above. For the original kernel we chose a RBF kernel with $\sigma = 1$ which is the value employed in the experiments reported in [12]. We used the kernelized version of the kernel design algorithm to learn a different kernel operator for each of the binary problems. We then used a variant of the Perceptron [11] and with the original RBF kernel and with the learned kernels. One of the motivations for using the Perceptron is its simplicity which can underscore differences in the kernels. We ran the kernel learning algorithm with LogLoss and ExpLoss, using bith the training set and the test test as $\tilde{S}$. Thus, we obtained four different sets of kernels where each set consists of $45$ kernels. By examining the training loss, we set the number of rounds of boosting to be $30$ for the LogLoss and $50$ for the ExpLoss, when using the trainin set. When using the test set, the number of rounds of boosting was set to $100$ for both losses. Since the algorithm exhibits slower rate of convergence with the test data, we choose a a higher value without attempting to optimize the actual value. The left plot of Fig. 5 is a scatter plot comparing the test error of each of the binary classifiers when trained with the original RBF a kernel versus the performance achieved on the same binary problem with a learned kernel. The kernels were built using boosting with the LogLoss and $\tilde{S}$ was the training data. In almost all of the $45$ binary classification problems, the learned kernels yielded lower error rates when combined with the Perceptron algorithm. The right plot of Fig. 5 compares two learned kernels: the first was build using the training instances as the templates constituing $\tilde{S}$ while the second used the test instances. Although the differenece between the two versions is not as significant as the difference on the left plot, we still achieve an overall improvement in about $25\%$ of the binary problems by using the test instances.

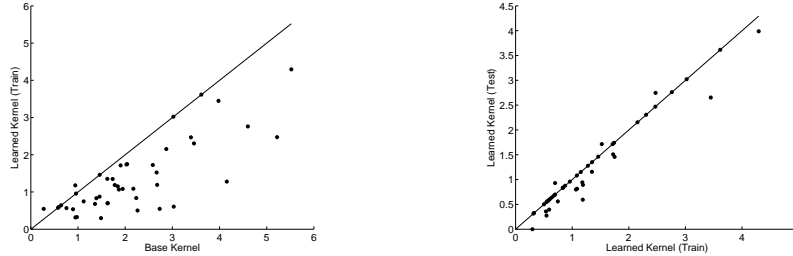

Figure 5: Left: a scatter plot comparing the error rate of $45$ binary classifiers trained using an RBF kernel ($x$-axis) and a learned kernel with training instances. Right: a similar scatter plot for a learned kernel only constructed from training instances ($x$-axis) and test instances.

## 5  Discussion

In this paper we showed how to use the boosting framework to design kernels. Our approach is especially appealing in transductive learning tasks where the test data distribution is different than the the distribution of the training data. For example, in speech recognition tasks the training data is often clean and well recorded while the test data often passes through a noisy channel that distorts the signal. An interesting and challanging question that stem from this research is how to extend the framework to accommodate more complex decision tasks such as multiclass and regression problems. Finally, we would like to note alternative approaches to the kernel design problem has been devised in parallel and independently. See [13, 14] for further details.

**Acknowledgements:** Special thanks to Cyril Goutte and to John Show-Taylor for pointing the connection to the generalized eigen vector problem. Thanks also to the anonymous reviewers for constructive comments.

## References

[1] V. N. Vapnik. *Statistical Learning Theory*. Wiley, 1998.

[2] N. Cristianini and J. Shawe-Taylor. *An Introduction to Support Vector Machines*. Cambridge University Press, 2000.

[3] Huma Lodhi, John Shawe-Taylor, Nello Cristianini, and Christopher J. C. H. Watkins. Text classification using string kernels. *Journal of Machine Learning Research*, 2:419–444, 2002.

[4] C. Leslie, E. Eskin, and W. Stafford Noble. The spectrum kernel: A string kernel for svm protein classification. In *Proceedings of the Pacific Symposium on Biocomputing*, 2002.

[5] Nello Cristianini, Andre Elisseeff, John Shawe-Taylor, and Jaz Kandla. On kernel target alignment. In *Advances in Neural Information Processing Systems 14*, 2001.

[6] G. Lanckriet, N. Cristianini, P. Bartlett, L. El Ghaoui, and M. Jordan. Learning the kernel matrix with semi-definite programming. In *Proc. of the 19th Intl. Conf. on Machine Learning*, 2002.

[7] Jerome Friedman, Trevor Hastie, and Robert Tibshirani. Additive logistic regression: a statistical view of boosting. *Annals of Statistics*, 28(2):337–374, April 2000.

[8] Michael Collins, Robert E. Schapire, and Yoram Singer. Logistic regression, adaboost and bregman distances. *Machine Learning*, 47(2/3):253–285, 2002.

[9] Llew Mason, Jonathan Baxter, Peter Bartlett, and Marcus Frean. Functional gradient techniques for combining hypotheses. In *Advances in Large Margin Classifiers*. MIT Press, 1999.

[10] Roger A. Horn and Charles R. Johnson. *Matrix Analysis*. Cambridge University Press, 1985.

[11] F. Rosenblatt. The perceptron: A probabilistic model for information storage and organization in the brain. *Psychological Review*, 65:386–407, 1958.

[12] B. Schölkopf, S. Mika, C.J.C. Burges, P. Knirsch, K. Müller, G. Rätsch, and A.J. Smola. Input space vs. feature space in kernel-based methods. *IEEE Trans. on NN*, 10(5):1000–1017, 1999.

[13] O. Bosquet and D.J.L. Herrmann. On the complexity of learning the kernel matrix. NIPS, 2002.

[14] C.S. Ong, A.J. Smola, and R.C. Williamson. Superkenels. NIPS, 2002.
